# Query By Committee Made Real

**Ran Gilad-Bachrach**[†◇]     **Amir Navot**[‡]     **Naftali Tishby**[†‡]
† School of Computer Science and Engineering
‡ Interdisciplinary Center for Neural Computation
The Hebrew University, Jerusalem, Israel.
◇ Intel Research

## Abstract

Training a learning algorithm is a costly task. A major goal of active learning is to reduce this cost. In this paper we introduce a new algorithm, KQBC, which is capable of actively learning large scale problems by using selective sampling. The algorithm overcomes the costly sampling step of the well known *Query By Committee* (QBC) algorithm by projecting onto a low dimensional space. KQBC also enables the use of kernels, providing a simple way of extending QBC to the non-linear scenario. Sampling the low dimension space is done using the *hit and run* random walk. We demonstrate the success of this novel algorithm by applying it to both artificial and a real world problems.

## 1   Introduction

Stone's celebrated theorem proves that given a large enough training sequence, even naive algorithms such as the $k$-nearest neighbors can be optimal. However, collecting large training sequences poses two main obstacles. First, collecting these sequences is a lengthy and costly task. Second, processing large datasets requires enormous resources. The selective sampling framework [1] suggests permitting the learner some control over the learning process. In this way, the learner can collect a short and informative training sequence. This is done by generating a large set of unlabeled instances and allowing the learner to select the instances to be labeled.

The *Query By Committee* algorithm (QBC) [2] was the inspiration behind many algorithms in the selective sampling framework [3, 4, 5]. QBC is a simple yet powerful algorithm. During learning it maintains a *version space*, the space of all the classifiers which are consistent with all the previous labeled instances. Whenever an unlabeled instance is available, QBC selects two random hypotheses from the version space and only queries for the label of the new instance if the two hypotheses disagree. Freund et al. [6] proved that when certain conditions apply, QBC will reach a generalization error of $\epsilon$ when using only $O\left(\log 1/\epsilon\right)$ labels. QBC works in an online fashion where each instance is considered only once to decide whether to query for its label or not. This is significant when there are a large number of unlabeled instances. In this scenario, batch processing of the data is unfeasible (see e.g. [7]). However, QBC was never implemented as is, since it requires the ability to sample hypotheses from the version space, a task that all known method do in an unreasonable amount of time [8].

The algorithm we present in this paper uses the same skeleton as QBC, but replaces sampling from the high dimensional version space by sampling from a low dimensional projection of it. By doing so, we obtain an algorithm which can cope with large scale problems and at the same time authorizes the use of kernels. Although the algorithm uses linear classifiers at its core, the use of kernels makes it much broader in scope. This new sampling method is presented in section 2. Section 3 gives a detailed description of the kernelized version, the *Kernel Query By Committee* (KQBC) algorithm. The last building block is a method for sampling from convex bodies. We suggest the *hit and run* [9] random walk for this purpose in section 4. A Matlab implementation of KQBC is available at `http://www.cs.huji.ac.il/labs/learning/code/qbc`.

The empirical part of this work is presented in section 5. We demonstrate how KQBC works on two binary classification tasks. The first is a synthetic linear classification task. The second involves differentiating male and female facial images. We show that in both cases, KQBC learns faster than Support Vector Machines (SVM) [10]. KQBC can be used to select a subsample to which SVM is applied. In our experiments, this method was superior to SVM; however, KQBC outperformed both.

**Related work:** Many algorithms for selective sampling have been suggested in the literature. However only a few of them have a theoretical justification. As already mentioned, QBC has a theoretical analysis. Two other notable algorithms are the greedy active learning algorithm [11] and the perceptron based active learning algorithm [12]. The greedy active learning algorithm has the remarkable property of being close to optimal in all settings. However, it operates in a batch setting, where selecting the next query point requires reevaluation of the whole set of unlabeled instances. This is problematic when the dataset is large. The perceptron based active learning algorithm, on the other hand, is extremely efficient in its computational requirements, but is restricted to linear classifiers since it requires the explicit use of the input dimension.

Graepel et al. [13] presented a *billiard walk* in the version space as a part of the Bayes Point Machine. Similar to the method presented here, the *billiard walk* is capable of sampling hypotheses from the version space when kernels are used. The method presented here has several advantages: it has better theoretical grounding and it is easier to implement.

## 2    A New Method for Sampling the Version-Space

The *Query By Committee* algorithm [2] provides a general framework that can be used with any concept class. Whenever a new instance is presented, QBC generates two independent predictions for its label by sampling two hypotheses from the version space[1]. If the two predictions differ, QBC queries for the label of the instance at hand (see algorithm 1). The main obstacle in implementing QBC is the need to sample from the version space (step 2b). It is not clear how to do this with reasonable computational complexity. As is the case for most research in machine learning, we first focus on the class of linear classifiers and then extend the discussion by using kernels. In the linear case, the dimension of the version space is the input dimension which is typically large for real world problems. Thus direct sampling is practically impossible. We overcome this obstacle by projecting the version space onto a low dimensional subspace.

Assume that the learner has seen the labeled sample $S = \{(x_i, y_i)\}_{i=1}^{k}$, where $x_i \in \mathbb{R}^d$ and $y_i \in \{\pm 1\}$. The version space is defined to be the set of all classifiers which correctly classify all the instances seen so far:

$$V = \{w \; : \; \|w\| \leq 1 \text{ and } \forall i \; y_i \, (w \cdot x_i) > 0\} \tag{1}$$

**Algorithm 1** Query By Committee [2]

**Inputs:**
- A concept class $\mathcal{C}$ and a probability measure $\nu$ defined over $\mathcal{C}$.

**The algorithm:**
1. Let $S \leftarrow \phi$, $V \leftarrow \mathcal{C}$.
2. For $t = 1, 2, \ldots$
   - (a) Receive an instance $x$.
   - (b) Let $h_1, h_2$ be two random hypotheses selected from $\nu$ restricted to $V$.
   - (c) If $h_1(x) \neq h_2(x)$ then
     - i. Ask for the label $y$ of $x$.
     - ii. Add the pair $(x, y)$ to $S$.
     - iii. Let $V \leftarrow \{c \in \mathcal{C} : \forall (x, y) \in S \ \ c(x) = y\}$.

---

QBC assumes a prior $\nu$ over the class of linear classifiers. The sample $S$ induces a posterior over the class of linear classifiers which is the restriction of $\nu$ to $V$. Thus, the probability that QBC will query for the label of an instance $x$ is exactly

$$2 \Pr_{w \sim \nu|V} [w \cdot x > 0] \Pr_{w \sim \nu|V} [w \cdot x < 0] \tag{2}$$

where $\nu|V$ is the restriction of $\nu$ to $V$.

From (2) we see that there is no need to explicitly select two random hypotheses. Instead, we can use any stochastic approach that will query for the label with the same probability as in (2). Furthermore, if we can sample $\hat{y} \in \{\pm 1\}$ such that

$$\Pr[\hat{y} = 1] = \Pr_{w \sim \nu|V}[w \cdot x > 0] \qquad \text{and} \tag{3}$$

$$\Pr[\hat{y} = -1] = \Pr_{w \sim \nu|V}[w \cdot x < 0] \tag{4}$$

we can use it instead, by querying the label of $x$ with a probability of $2 \Pr[\hat{y} = 1] \Pr[\hat{y} = -1]$. Based on this observation, we introduce a stochastic algorithm which returns $\hat{y}$ with probabilities as specified in (3) and (4). This procedure can replace the sampling step in the QBC algorithm.

Let $S = \{(x_i, y_i)\}_{i=1}^{k}$ be a labeled sample. Let $x$ be an instance for which we need to decide whether to query for its label or not. We denote by $V$ the version space as defined in (1) and denote by $T$ the space spanned by $x_1, \ldots, x_k$ and $x$. QBC asks for two random hypotheses from $V$ and queries for the label of $x$ only if these two hypotheses predict different labels for $x$. Our procedure does the same thing, but instead of sampling the hypotheses from $V$ we sample them from $V \cap T$. One main advantage of this new procedure is that it samples from a space of low dimension and therefore its computational complexity is much lower. This is true since $T$ is a space of dimension $k + 1$ at most, where $k$ is the number of queries for label QBC made so far. Hence, the body $V \cap T$ is a low-dimensional convex body[2] and thus sampling from it can be done efficiently. The input dimension plays a minor role in the sampling algorithm. Another important advantage is that it allows us to use kernels, and therefore gives a systematic way to extend QBC to the non-linear scenario. The use of kernels is described in detail in section 3.

The following theorem proves that indeed sampling from $V \cap T$ produces the desired results. It shows that if the prior $\nu$ (see algorithm 1) is uniform, then sampling hypotheses uniformly from $V$ or from $V \cap T$ generates the same results.

**Theorem 1** *Let $S = \{(x_i, y_i)\}_{i=1}^k$ be a labeled sample and $x$ an instance. Let the version space be $V = \{w : \|w\| \leq 1 \text{ and } \forall i \ y_i (w \cdot x_i) > 0\}$ and $T = \text{span}(x, x_1, \ldots, x_k)$ then*

$$\Pr_{w \sim U(V)} [w \cdot x > 0] = \Pr_{w \sim U(V \cap T)} [w \cdot x > 0] \qquad and$$
$$\Pr_{w \sim U(V)} [w \cdot x < 0] = \Pr_{w \sim U(V \cap T)} [w \cdot x < 0]$$

*where $U(\cdot)$ is the uniform distribution.*

The proof of this theorem is given in the supplementary material [14].

## 3  Sampling with Kernels

In this section we show how the new sampling method presented in section 2 can be used together with kernel. QBC uses the random hypotheses for one purpose alone: to check the labels they predict for instances. In our new sampling method the hypotheses are sampled from $V \cap T$, where $T = \text{span}(x, x_1, \ldots, x_k)$. Hence, any hypothesis is represented by $w \in V \cap T$, that has the form

$$w = \alpha_0 x + \sum_{j=1}^k \alpha_j x_j \tag{5}$$

The label $w$ assigns to an instance $x'$ is

$$w \cdot x' = \left( \alpha_0 x + \sum_{j=1}^k \alpha_j x_j \right) \cdot x' = \alpha_0 x \cdot x' + \sum_{j=1}^k \alpha_j x_j \cdot x' \tag{6}$$

Note that in (6) only inner products are used, hence we can use kernels. Using these observations, we can sample a hypothesis by sampling $\alpha_0, \ldots, \alpha_k$ and define $w$ as in (5). However, since the $x_i$'s do not form an orthonormal basis of $T$, sampling the $\alpha$'s uniformly is not equivalent to sampling the $w$'s uniformly. We overcome this problem by using an orthonormal basis of $T$. The following lemma shows a possible way in which the orthonormal basis for $T$ can be computed when only inner products are used. The method presented here does not make use of the fact that we can build this basis incrementally.

**Lemma 1** *Let $x_0, \ldots, x_k$ be a set of vectors, let $T = \text{span}(x_0, \ldots, x_k)$ and let $G = (g_{i,j})$ be the Grahm matrix such that $g_{i,j} = x_i \cdot x_j$. Let $\lambda_1, \ldots, \lambda_r$ be the non-zero eigen values of $G$ with the corresponding eigen-vectors $\gamma_1, \ldots, \gamma_r$. Then the vectors $t_1, \ldots, t_r$ such that*

$$t_i = \sum_{l=0}^k \frac{\gamma_i(l)}{\sqrt{\lambda_i}} x_l$$

*form an orthonormal basis of the space $T$.*

The proof of lemma 1 is given in the supplementary material [14]. This lemma is significant since the basis $t_1, \ldots, t_r$ enables us to sample from $V \cap T$ using simple techniques. Note that a vector $w \in T$ can be expressed as $\sum_{i=1}^r \alpha(i) t_i$. Since the $t_i$'s form an orthonormal basis, $\|w\| = \|\alpha\|$. Furthermore, we can check the label $w$ assigns to $x_j$ by

$$w \cdot x_j = \sum_i \alpha(i) t_i \cdot x_j = \sum_{i,l} \alpha(i) \frac{\gamma_i(l)}{\sqrt{\gamma_i}} x_l \cdot x_j$$

which is a function of the Grahm matrix. Therefore, sampling from $V \cap T$ boils down to the problem of sampling from convex bodies, where instead of sampling a vector directly we sample the coefficients of the orthonormal basis $t_1, \ldots, t_r$.

There are several methods for generating the final hypothesis to be used in the generalization phase. In the experiments reported in section 5 we have randomly selected a single hypothesis from $V \cap T$ and used it to make all predictions, where $V$ is the version space at the time when the learning terminated and $T$ is the span of all instances for which KQBC queried for label during the learning process.

## 4  Hit and Run

Hit and run [9] is a method of sampling from a convex body $\mathcal{K}$ using a random walk. Let $z \in \mathcal{K}$, a single step of the hit and run begins by choosing a random point $u$ from the unit sphere. Afterwards the algorithm moves to a random point selected uniformly from $l \cap \mathcal{K}$, where $l$ is the line passing through $z$ and $z + u$.

Hit and run has several advantages over other random walks for sampling from convex bodies. First, its stationary distribution is indeed the uniform distribution, it mixes fast [9] and it does not require a "warm" starting point [15]. What makes it especially suitable for practical use is the fact that it does not require any parameter tuning other than the number of random steps. It is also very easy to implement.

Current proofs [9, 15] show that $O^* \left( d^3 \right)$ steps are needed for the random walk to mix. However, the constants in these bounds are very large. In practice hit and run mixes much faster than that. We have used it to sample from the body $V \cap T$. The number of steps we used was very small, ranging from couple of hundred to a couple of thousands. Our empirical study shows that this suffices to obtain impressive results.

## 5  Empirical Study

In this section we present the results of applying our new kernelized version of the query by committee (KQBC), to two learning tasks. The first task requires classification of synthetic data while the second is a real world problem.

### 5.1  Synthetic Data

In our first experiment we studied the task of learning a linear classifier in a $d$-dimensional space. The target classifier is the vector $w^* = (1, 0, \ldots, 0)$ thus the label of an instance $x \in \mathbb{R}^d$ is the sign of its first coordinate. The instances were normally distributed $N \left( \mu = 0, \Sigma = I_d \right)$. In each trial we used 10000 unlabeled instances and let KQBC select the instances to query for the labels. We also applied Support Vector Machine (SVM) to the same data in order to demonstrate the benefit of using active learning. The linear kernel was used for both KQBC and SVM. Since SVM is a passive learner, SVM was trained on prefixes of the training data of different sizes. The results are presented in figure 1.

The difference between KQBC and SVM is notable. When both are applied to a 15-dimensional linear discrimination problem (figure 1b), SVM and KQBC have an error rate of $\sim 6\%$ and $\sim 0.7\%$ respectively after 120 labels. After such a short training sequence the difference is of an order of magnitude. The same qualitative results appear for all problem sizes.

As expected, the generalization error of KQBC decreases exponentially fast as the number of queries is increased, whereas the generalization error of SVM decreases only at an inverse-polynomial rate (the rate is $O^* \left( 1/k \right)$ where $k$ is the number of labels). This should not come as a surprise since Freund et al. [6] proved that this is the expected behavior. Note also that the bound of $50 \cdot 2^{-0.67k/d}$ over the generalization error that was proved in [6] was replicated in our experiments (figure 1c).

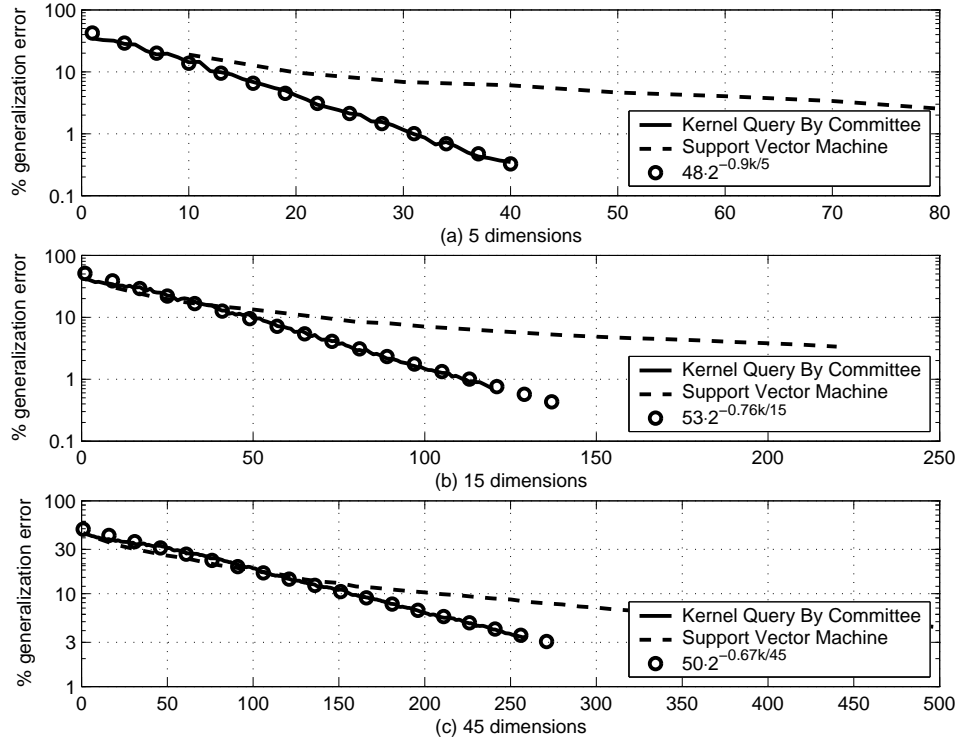

Figure 1: **Results on the synthetic data**. The generalization error (y-axis) in percents (in logarithmic scale) versus the number of queries (x-axis). Plots (a), (b) and (c) represent the synthetic task in 5, 15 and 45 dimensional spaces respectively. The generalization error of KQBC is compared to the generalization error of SVM. The results presented here are averaged over 50 trials. Note that the error rate of KQBC decreases exponentially fast. Recall that [6] proved a bound on the generalization error of $50 \cdot 2^{-0.67k/d}$ where $k$ is the number of queries and $d$ is the dimension.

## 5.2  Face Images Classification

The learning algorithm was then applied in a more realistic setting. In the second task we used the AR face images dataset [16]. The people in these images are wearing different accessories, have different facial expressions and the faces are lit from different directions. We selected a subset of $1456$ images from this dataset. Each image was converted into gray-scale and re-sized to $85 \times 60$ pixels, i.e. each image was represented as a $5100$ dimensional vector. See figure 2 for sample images. The task was to distinguish male and female images. For this purpose we split the data into a training sequence of $1000$ images and a test sequence of $456$ images. To test statistical significance we repeated this process $20$ times, each time splitting the dataset into training and testing sequences.

We applied both KQBC and SVM to this dataset. We used the Gaussian kernel: $K\left(x_1, x_2\right) = \exp\left(-\left\|x_1 - x_2\right\|^2 / 2\sigma^2\right)$ where $\sigma = 3500$ which is the value favorable by SVM. The results are presented in figure 3. It is apparent from figure 3 that KQBC outperforms SVM. When the budget allows for $100 - 140$ labels, KQBC has an error rate of $2 - 3$ percent less than the error rate of SVM. When $140$ labels are used, KQBC outperforms SVM by $3.6\%$ on average. This difference is significant as in $90\%$ of the trials

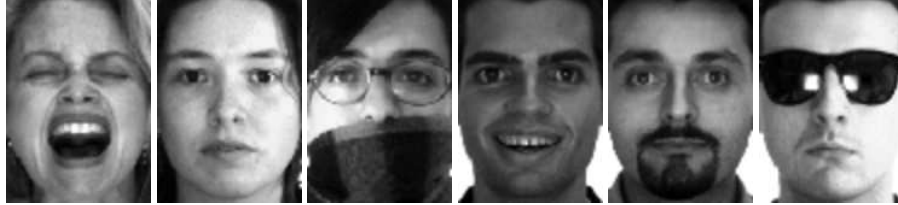

Figure 2: **Examples of face images used for the face recognition task**.

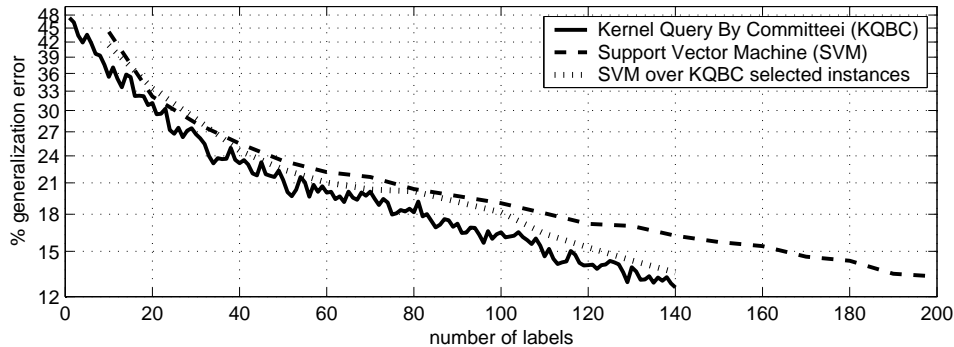

Figure 3: **The generalization error of KQBC and SVM for the faces dataset** (averaged over 20 trials). The generalization error (y-axis) vs. number of queries (x-axis) for KQBC (solid) and SVM (dashed) are compared. When SVM was applied solely to the instances selected by KQBC (dotted line) the results are better than SVM but worse than KQBC.

KQBC outperformed SVM by more than $1\%$. In one of the cases, KQBC was $11\%$ better.

We also used KQBC as an active selection method for SVM. We trained SVM over the instances selected by KQBC. The generalization error obtained by this combined scheme was better than the passive SVM but worse than KQBC.

In figure 4 we see the last images for which KQBC queried for labels. It is apparent, that the selection made by KQBC is non-trivial. All the images are either highly saturated or partly covered by scarves and sunglasses. We conclude that KQBC indeed performs well even when kernels are used.

## 6 Summary and Further Study

In this paper we present a novel version of the QBC algorithm. This novel version is both efficient and rigorous. The time-complexity of our algorithm depends solely on the number of queries made and not on the input dimension or the VC-dimension of the class. Furthermore, our technique only requires inner products of the labeled data points - thus it can be implemented with kernels as well.

We showed a practical implementation of QBC using kernels and the hit and run random walk which is very close to the "provable" version. We conducted a couple of experiments with this novel algorithm. In all our experiments, KQBC outperformed SVM significantly. However, this experimental study needs to be extended. In the future, we would like to compare our algorithm with other active learning algorithms, over a variety of datasets.

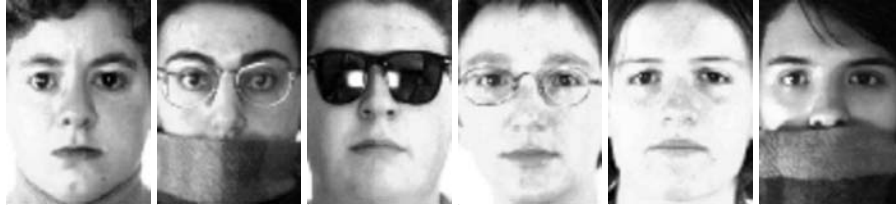

Figure 4: **Images selected by KQBC**. The last six faces for which KQBC queried for a label. Note that three of the images are saturated and that two of these are wearing a scarf that covers half of their faces.

## Footnotes

[1]The version space is the collection of hypotheses that are consistent with previous labels.

[2]From the definition of the version space $V$ it follows that it is a convex body.

# References

[1] D. Cohn, L. Atlas, and R. Ladner. Training connectionist networks with queries and selective sampling. *Advanced in Neural Information Processing Systems 2*, 1990.

[2] H. S. Seung, M. Opper, and H. Sompolinsky. Query by committee. *Proc. of the Fifth Workshop on Computational Learning Theory*, pages 287–294, 1992.

[3] C. Campbell, N. Cristianini, and A. Smola. Query learning with large margin classifiers. In *Proc. 17'th International Conference on Machine Learning (ICML)*, 2000.

[4] S. Tong. *Active Learning: Theory and Applications*. PhD thesis, Stanford University, 2001.

[5] G. Tur, R. Schapire, and D. Hakkani-Tür. Active learning for spoken language understanding. In *Proc. IEEE International Conference on Acoustics, Speech and Signal Processing*, 2003.

[6] Y. Freund, H. Seung, E. Shamir, and N. Tishby. Selective sampling using the query by committee algorithm. *Machine Learning*, 28:133–168, 1997.

[7] H. Mamitsuka and N. Abe. Efficient data mining by active learning. In *Progress in Discovery Science*, pages 258–267, 2002.

[8] R. Bachrach, S. Fine, and E. Shamir. Query by committee, linear separation and random walks. *Theoretical Computer Science*, 284(1), 2002.

[9] L. Lovász and S. Vempala. Hit and run is fast and fun. Technical Report MSR-TR-2003-05, Microsoft Research, 2003.

[10] B. Boser, I. Guyon, and V. Vapnik. Optimal margin classifiers. In *Fifth Annual Workshop on Computational Learning Theory*, pages 144–152, 1992.

[11] S. Dasgupta. Analysis of a greedy active learning strategy. In *Neural Information Processing Systems (NIPS)*, 2004.

[12] S. Dasgupta, A. T. Kalai, and C. Monteleoni. Analysis of perceptron-based active learning. In *Proceeding of the 18th annual Conference on Learning Theory (COLT)*, 2005.

[13] R. Herbrich, T. Graepel, and C. Campbell. Bayes point machines. *Journal of Machine Learning Research*, 1:245–279, 2001.

[14] R. Gilad-Bachrach, A. Navot, and N. Tishby. Query by committee made real - supplementary material. http://www.cs.huji.ac.il/∼ranb/kqcb_supp.ps.

[15] L. Lovász and S. Vempala. Hit-and-run from a corner. In *Proc. of the 36th ACM Symposium on the Theory of Computing (STOC)*, 2004.

[16] A.M. Martinez and R. Benavente. The ar face database. Technical report, CVC Tech. Rep. #24, 1998.
